# An ideal observer model for identifying the reference frame of objects

**Joseph L. Austerweil**
Department of Psychology
University of California, Berkeley
Berkeley, CA 94720
Joseph.Austerweil@gmail.com

**Abram L. Friesen**
Department of Computer Science and Engineering
University of Washington
Seattle, WA 98195
afriesen@cs.washington.edu

**Thomas L. Griffiths**
Department of Psychology
University of California, Berkeley
Berkeley, CA 94720
Tom_Griffiths@berkeley.edu

## Abstract

The object people perceive in an image can depend on its orientation relative to the scene it is in (its reference frame). For example, the images of the symbols $\times$ and $+$ differ by a 45 degree rotation. Although real scenes have multiple images and reference frames, psychologists have focused on scenes with only one reference frame. We propose an ideal observer model based on nonparametric Bayesian statistics for inferring the number of reference frames in a scene and their parameters. When an ambiguous image could be assigned to two conflicting reference frames, the model predicts two factors should influence the reference frame inferred for the image: The image should be more likely to share the reference frame of the closer object (*proximity*) and it should be more likely to share the reference frame containing the most objects (*alignment*). We confirm people use both cues using a novel methodology that allows for easy testing of human reference frame inference.

## 1 Introduction

When are the objects in two images the same?[1] Although people recognize and categorize objects successfully and effortlessly, object recognition in machine learning is an incredibly difficult problem and people's success is a puzzle to cognitive scientists. To solve this problem, object recognition techniques typically generate a set of features using a predefined procedure (e.g., SIFT descriptors [1] or textons [2]) or learn features (e.g., deep belief networks [3]) from the images. The general goal of these methods is to extract features from images that are useful for identifying the objects that generated the images after whatever transformations occurred while producing them (e.g., viewpoint changes). This is a sensible strategy given that people typically perceive the same object even when it is transformed in its image (e.g., translations). However, not all transformations should be ignored: The perceived identity of some objects depends on the orientation of its features with respect to the scene it is in (e.g., $\times$ vs. $+$ differ only in orientation), but for other objects it does

not. Developing proper object recognition and fully understanding how people do it depends on explaining how people determine the orientation of objects with respect to the scene they are in.

The importance of orientation for object recognition leads us to the following question: If two objects project to the same image under different viewing conditions (e.g., $+$ and $\times$ after 45 degree rotations), how do people infer which object is in the image? In psychology, there are two main theories for how people solve this problem: the *invariant feature* hypothesis [4], which is essentially the strategy taken by current object recognition techniques (use features that preserve object identity over the possible transformations that generate images of the object), and the *reference frame* hypothesis, which posits that objects are embedded in coordinate axes [5]. The coordinate axes set the orientation and scale of the objects, and thus $+$ and $\times$ can be identified as different objects. Though they may produce the same image, they will have different coordinate axes.

In some situations the orientation of an image's reference frame is simply the orientation of the retina; however, this is not the case when we rotate our heads (as our retinal image rotates) or look at a rotated object (e.g., a person lying on a bench or a document rotated on a desk). Thus, the reference frame of an image is ambiguous without additional information. However, if there is another object in the scene whose orientation is unambiguous (like a 5), then the orientation of the ambiguous image can be inferred.[2] We demonstrate that people use the orientation of other images in the scene to determine the orientation of an ambiguous image by asking participants to solve arithmetic problems, where the operator image is ambiguous and the two numbers flanking the operator are either oriented upright or rotated 45 degrees. The solution people adopt is indicative of the reference frame they inferred for the operator (multiplication implies an upright reference frame and addition implies a diagonal reference frame). This is a novel experimental method that allows us to explore reference frame inference in a wide range of contexts.

In real life, we typically view scenes with multiple reference frames. For example, some books on a bookshelf might be upright, other books could be tilted diagonally (for support), while other books might lie flat. Yet there has been little work investigating how people infer the number of reference frames, their orientations, and which images belong to each reference frame. To solve this problem, we note that each image in a scene belongs to a single reference frame, and thus reference frames form a partition of the images in a scene (where each block in the partition corresponds to a reference frame). Using a standard nonparametric Bayesian model for partitions, we formulate an ideal observer model to infer multiple reference frames and their parameters. The model predicts that people should be sensitive to two cues when inferring the reference frames of a scene: the *proximity* of the ambiguous image to two unambiguous flanking images in conflicting orientations, and the difference in the number of objects *aligned* in the competing reference frames. We confirm people are sensitive to both cues using the novel method described above.

The summary of the article is as follows. First, Section 2 summarizes relevant psychological research on how orientation affects the objects perceived in ambiguous images. Next, Section 3 develops a novel method for online testing of the reference frame people infer for an image and establishes its efficacy. Section 4 presents an ideal observer model for reference frame inference in scenes with multiple reference frames. The model predicts that the ambiguous image's *proximity* to other reference frames should affect the inferred reference frame and Section 5 confirms that people act in accordance with this prediction in a behavioral experiment. The model also predicts that the number of *aligned* objects in a reference frame should affect the reference frame inferred for an ambiguous image. Section 6 confirms this prediction in a behavioral experiment. Section 7 concludes the paper and highlights some directions for future research.

## 2  Orientation in psychological theories of object representation

Though the perceived object of some images does not depend on its orientation (like a 5), there are many examples where the perceived object does depend on its orientation [7, 8], including $+$ vs. $\times$ or a square vs. a diamond, and other effects of orientation on object recognition [9, 10]. This has led psychologists to believe that people represent objects within a reference frame (a set of coordinate axes).[3] Figure 1 (a) shows that reference frames predict the image $+$ is interpreted as a $+$ when

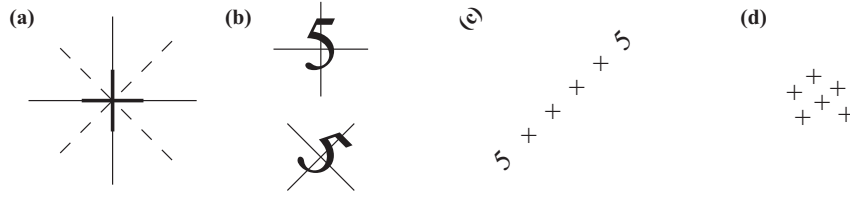

Figure 1: Reference frames. (a) The ambiguity of the + image can be resolved using reference frames: a + with horizontal orientation (solid axes) or a × rotated 45 degrees (dashed axes). (b) Other images are unambiguous, like a 5. (c) The reference frame of ambiguous objects is influenced by objects with unambiguous reference frames. (d) The group of objects is seen as either all + or all ×, but not some + and some ×. This establishes one reference frame per group.

the coordinate axes are aligned with the document's axes and as × when the coordinate axes are diagonal to the document's axes. For objects that are rotationally invariant, there is only one object that generates the observed image and so it is identifiable in any orientation (see Figure 1 (b)). The dependence of object perception on orientation is a well established norm and has been demonstrated with novel and familiar 2-D objects, faces, handwriting [8, 9], and 3-D objects [10, 11].

Central to the reference frame hypothesis is the ability of our perceptual system to infer a reference frame for a given image. As more than one reference frame may be consistent with an observed image, psychologists have explored how people infer the appropriate reference frame for an image. Though reference frame inference is strongly influenced by the top-down axis of the retinal image and by the axis of gravity (given by our proprioceptive and vestibular senses) [8], the scene itself can influence the inferred reference frame. Objects grouped together in the world tend to be affected by the same transformation when they generate images (e.g., the text on a poster as the poster is rotated), and so it is sensible that the inferred reference frame for an ambiguous image is influenced by the orientations of the images surrounding it. Figures 1 (c) and (d) are phenomenological demonstrations of how the *alignment* of the orientations of other objects in a scene can bias the inferred reference frame for an image whose reference frame is ambiguous (and there is strong corroborating empirical evidence for this principle [12, 13]).[4] Figure 1 (c) is biased towards being interpreted as × based on the surrounding context and the images in Figure 1 (d) are interpreted as either all + or all tilted ×, but it is difficult to interpret some as + and others as tilted × simultaneously [14]. Thus, there is one reference frame shared by all the objects in a group.

Although there is a wealth of research into reference frame inference for scenes containing a single reference frame, to the best of our knowledge, there has not been any research into how people determine the reference frame of ambiguously oriented images when there is more than one reference frame in the scene (and both are consistent with the images). Before exploring what cues influence human reference frame inference in scenes with multiple reference frames, we develop a novel method for testing human reference frame inference.

## 3   Testing reference frame inference using arithmetic

To test how different factors influence the reference frame people infer for an image, we ask people to solve an arithmetic problem without specifying the appropriate operation. If people view × and their response is the multiplication answer, then their reference frame for × is aligned with the horizontal and vertical axes of the page. Alternatively, if people view the same ×, but their response is the addition answer, then their reference frame for × is aligned with the axes diagonal to the page (and thus, relative to its own reference frame, it is treated as +).[5] We use this new method instead of previous techniques (e.g., explicitly asking the image's orientation and recording the frequency each orientation is chosen that is either compatible or conflicting to the tested hypothesis [15]) due to its ability to be used in a wide range of contexts and to demonstrate the robust importance of reference

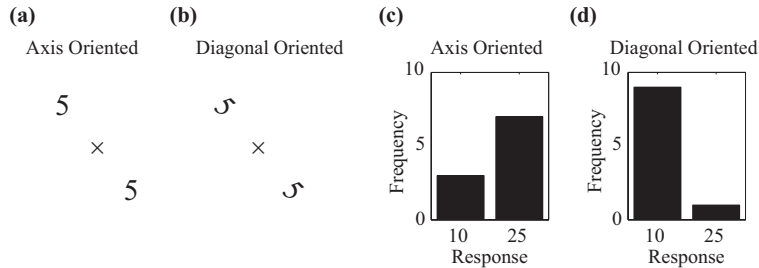

Figure 2: Effect of the orientations of other objects in the same reference frame. (a) 5s aligned with axes implies that the operator is $\times$. (b) 5s aligned with diagonal implies the operator is $+$ at a diagonal orientation. (c) Frequency of answers to (a) given by participants. Most participants respond with 25, the solution to the product of 5 and 5, meaning their reference frame is aligned with the axes of the page. (d) Frequency of answers to (b) given by participants. Most participants respond with 10, meaning their reference frame is aligned with the diagonals of the page.

frame inference on a seemingly unrelated cognitive behavior (solving an arithmetic problem). We confirm its validity by reproducing a previously found effect – the influence of orientation on other images in the scene [12].

When the reference frame for an image is ambiguous, one factor that influences the inferred reference frame is the orientation of other images it is grouped with, especially when those images are identifiable in any orientation. Thus, if we ask people to solve an arithmetic problem, where the operator $\times$ is paired with the numbers 5 aligned with the top-down axes of the page (Figure 2 (a)), they should respond 25, the result of multiplication. Alternatively, if people solve the same problem except the numbers 5 are aligned diagonally, they should infer the diagonal axes to be the reference frame and respond 10, the result of addition (Figure 2 (b)).

To test this method, we recruited 20 participants online, who answered one arithmetic problem in exchange for a small monetary reward. The participants were counterbalanced over the axis or diagonally oriented conditions (Figures 2 (a) and (b) respectively) and all participants gave either the addition (10) or multiplication (25) solution. By changing the orientation of the numbers, the solutions to the arithmetic problems given by participants in Figures 2 (a) and (b) are different despite having identical numbers and the identical operator image. Figures 2 (c) and (d) show that the responses of two groups of participants who answered the arithmetic problem in (a) and (b) differed as predicted ($\chi^2(1) = 5.208, p < 0.05$, using Yates' chi-square correction). Thus, asking participants to solve arithmetic problems is an effective method for testing reference frame inference and perceived orientations can influence higher level cognition.

## 4 Modeling reference frame inference

Before describing our model of reference frame inference with multiple reference frames, we first present a probabilistic model for scenes of multiple images with only a single reference frame.

### 4.1 Reference frame inference for scenes with one reference frame

We assume that a vocabulary of possible objects is known ahead of time of size $V$ and that there are $R$ possible rotations. Each scene (e.g., Figure 2 (a) is one scene) consists of a set of images (e.g., 5, $\times$, and 5 are the images of Figure 2 (a)). For each image $i$ in a scene, the model is given its visual properties $\mathbf{y}_i$ and its spatial location $\mathbf{x}_i = (x_{i1}, x_{i2})$ The visual properties of the image $\mathbf{y}_i$ are generated by an unknown object $v_i$ rotated by $r$, the orientation of the scene's reference frame. A $V \times R$ binary image-object alignment matrix $\mathbf{A}^{(i)}$ encodes the object-rotation pairs consistent with the observed image $\mathbf{y}_i$ such that $\mathbf{A}^{(i)}(v, r) = 1$ if the image of object $v$ rotated $r$ degrees is consistent with $\mathbf{y}_i$. The model assumes that the spatial locations of the images are independent identically distributed draws from a Gaussian distribution with shared parameters $\mu$, the center point for the reference frame, and $\mathbf{\Sigma}$, the spread of objects around its center point. The unobserved objects and the orientation of the reference frame $r$ are drawn from independent discrete distributions

with parameters $\phi$ and $\theta$, the prior over objects and reference frame orientations, respectively. The following generative model defines our statistical model:

$$r|\theta \sim \text{Discrete}(\theta) \qquad\qquad v_i|\phi \overset{iid}{\sim} \text{Discrete}(\phi)$$

$$\mathbf{x}_i|\mu, \mathbf{\Sigma} \sim \text{Gaussian}\,(\mu, \mathbf{\Sigma}) \qquad\qquad P(\mathbf{y}_i|v_i, r) = \mathbf{A}^{(i)}(v_i, r)$$

If the model assumes there are three types of objects (5, $+$ and $\times$) and two possible rotations (0 and 45 degrees), the model captures the sensitivity of participants in the demonstration (Figure 2). In Figure 2 (a), the 5s are oriented at 0 degrees. $\mathbf{A}(5, r)$ is only non-zero when $r = 0$ because no other object can produce an image consistent with the observed image of the 5. $r = 0$ implies that the operator is $\times$, which is consistent with participant responses (Figure 2 (c)). When the 5s are oriented at 45 degrees (Figure 2 (b)), $\mathbf{A}(5, r)$ is only non-zero when $r = 45$ for the same reason as before. $r = 45$ implies that the operator is $+$, which is consistent with participant responses (Figure 2 (d)).

## 4.2 Extending the model for scenes with multiple reference frames

Although the model defined in the previous section succeeds in inferring the reference frame of an ambiguous image using other images it is grouped with, it cannot handle scenes containing multiple reference frames, such as the scenes in Figure 3. We extend the model by partitioning the images of a scene into reference frames, where each image of the scene belongs to exactly one reference frame and a reference frame is a block of the partition. From this perspective, inferring multiple reference frames for a scene of images is equivalent to partitioning the scene or clustering the images.

With the insight that grouping images into reference frames is like finding a partition of a scene, we can extend our model to select the reference frames of a scene (with an unknown number of reference frames). First, we generate a partition of the images in the scene from the Chinese restaurant process (CRP) [16] with parameter $\alpha$, an exchangeable distribution over partitions. The CRP is defined through the following sequential construction:

$$P(c_i = k|c_1, \ldots, c_{i-1}) = \begin{cases} \frac{n_k}{\alpha+i-1} & k \leq K \\ \frac{\alpha}{\alpha+i-1} & k = K+1 \end{cases}$$

where $K$ is the current number of reference frames and $n_k$ is the number of objects assigned to reference frame $k$. $c_i$ denotes the reference frame that object $i$ is assigned to and if $c_i = K + 1$, it is assigned a new reference frame containing none of the previous objects and $K$ increments by one (to initialize, the first object starts its own reference frame and $K = 1$). This gives us an assignment vector $\mathbf{c}$, where $c_i = j$ denotes reference frame $j$ contains image $i$. Each block in the partition (reference frame) $j$ is associated with a rotation $r_j$ and is embedded in the spatial layout of the scene with a center position $\mu_j$ and spread $\mathbf{\Sigma}_j$ (each of which is generated from a Gaussian-Inverse Wishart distribution with shared parameters). Thus, we have defined the following generative model for a set of images in a scene:

$$\mathbf{c}|\alpha \sim \text{CRP}(\alpha) \qquad\qquad \mu_j, \mathbf{\Sigma}_j|\mu_0, \mathbf{\Sigma}_0, k_0, \nu_0 \overset{iid}{\sim} \text{GIW}\,(\mu_0, \mathbf{\Sigma}_0, k_0, \nu_0)$$

$$r_j|\theta \overset{iid}{\sim} \text{Discrete}(\theta) \qquad\qquad v_i|\phi \overset{iid}{\sim} \text{Discrete}(\phi)$$

$$\mathbf{x}_i|c_i, \mu_{c_i}, \mathbf{\Sigma}_{c_i} \sim \text{Gaussian}\,(\mu_{c_i}, \mathbf{\Sigma}_{c_i}) \qquad\qquad P(\mathbf{y}_i|v_i, r_{c_i}, c_i) = \mathbf{A}^{(i)}(v_i, r_{c_i})$$

where GIW signifies the Gaussian-Inverse-Wishart distribution, and $\alpha$, $\mu_0$, $\mathbf{\Sigma}_0$, $k_0$, $\nu_0$, $\theta$, and $\phi$ are hyperparameters of our model.

We use Gibbs sampling for inference [17], which gives us the cluster assignments for each image and the updated parameters $\psi_j = (\mu_j, \mathbf{\Sigma}_j, r_j)$ for each cluster $j$. We begin by assigning each image to its own reference frame and then iterating. For each observed image, we resample $c_i$ from the set of existing clusters and $m = 2$ newly drawn clusters. After all $c_i$ values have been resampled, we discard any empty clusters and update the parameters of the remaining clusters by drawing them from their posterior distribution given the objects assigned to that reference frame $p(\psi_j|\{\mathbf{x}_i, \mathbf{y}_i : c_i = j\})$, where $\{\mathbf{x}_i, \mathbf{y}_i : c_i = j\}$ is the set of images and their locations in reference frame $j$.

## 4.3 Predictions for human reference frame inference

What factors influence the reference frame assigned to an ambiguous image according to our ideal observer model? Two factors it predicts should influence the image's inferred reference frame are

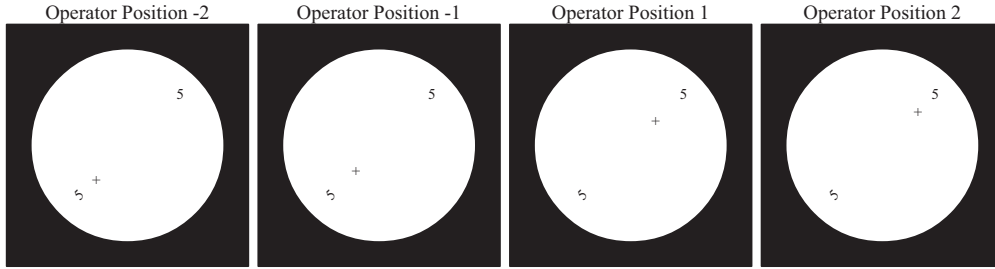

Figure 3: Trials from Experiment 1 showing the possible positions of the operators for the main factor of the experiment. Other factors randomized over trials are the numbers in the problem (always single digits), which of the two numbers was rotated, the diagonal that the numbers and operator are aligned on (positive diagonal shown in the figure, but numbers and operator aligned on the negative diagonal as well), and the rotation of the operator.

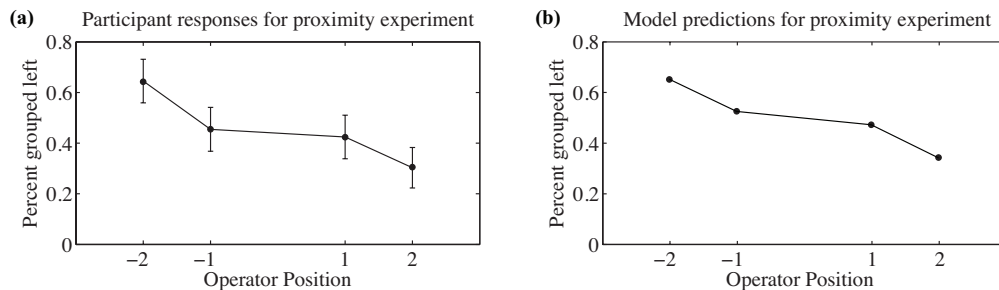

Figure 4: Proximity effects: (a) Human results and (b) Model results. The closer the operator is to the left number, the more likely it is to take the left number's orientation.

*proximity* or how close the image is to unambiguous images (as images in the same reference frame are coupled in spatial location) and *alignment* or the difference in the number of images assigned to each reference frame. The general paradigm we use to test the predictions is to have the + or × operator flanked by a number with different orientations on each side (see examples in Figure 3). It is clear that the two numbers should have their own reference frame, but it is ambiguous which reference frame the operator should be assigned to. We compare how each of these factors influences the reference frames inferred in the scene by people and our model in two behavioral experiments.

## 5   Experiment 1: Proximity effects on reference frame inference

When the reference frame for an image is ambiguous and there are two conflicting neighboring reference frames, our model predicts that *proximity* or the distance of the ambiguous image to the two conflicting reference frames should affect the reference frame adopted by the ambiguous image. We explore this question using the method presented above, where participants are asked to solve an arithmetic problem where the operator is ambiguous between + or × and the two numbers have conflicting reference frames (orientations). This allows us to deduce the reference frame inferred for the operator image from the answer given by participants. We manipulate proximity by changing the location of the operator such that it is closer to one of the two numbers as shown in Figure 3.

### 5.1   Methods

A total of 134 participants completed the experiment online through Amazon Mechanical Turk in exchange for \$0.20 USD. Four participants did not give a correct solution to the arithmetic problem (neither the addition nor multiplication solution) leaving 130 participants for analysis. Participants were asked to maximize their window before answering the arithmetic problem. All factors were manipulated between subjects as preliminary testing demonstrated a strong effect of trial order on the selected reference frame (probably because reference frames rarely change in the world).

The primary factor of interest of the experiment was the position of the operator scored from -2 (far to the left) to 2 (far to the right), which was counterbalanced over participants (without the 0 position). The problem was viewed through a simulated aperture (to minimize the effect of the monitor's reference frame). See Figure 3 for example trials with the operator in each position. There were several other factors that were randomized over participants: the numbers in the problem (randomly chosen single digit numbers), which number was rotated (left or right), the diagonal that the numbers and operator were aligned on (positive diagonal, as shown in Figure 3, or negative diagonal), and the rotation of the operator ($+$ or $\times$).

### 5.2 Results and Discussion

Figure 4 (a) shows that participants are more likely to infer the orientation of the left number for the operator the closer it is to the left number. The results confirm our hypothesis: the closer the operator is to an image with an unambiguous reference frame, the more likely participants are to infer that reference frame for the operator ($\chi^2(1) = 3.99, p < 0.05$ for -2 vs. 2). A probit regression analysis corroborates this result as the regression coefficient is significantly different from zero ($p < 0.05$).

The model results were generated using Gibbs sampling (as previously described) and shown in Figure 4 (b). For each trial, we ran the sampler for 50 burn-in iterations, recorded 750 samples, and then thinned the samples by selecting every 5 samples. This left 150 samples that formed our estimate for the proportion of times the operator grouped with the left reference frame. The parameters were initialized to: $\alpha = 0.001$, $\mu_0 = [264.7, 261.94]$, $\Sigma_0 = 1000\mathbf{I}$ (scenes are $550 \times 550$ pixels with the bottom-left corner as origin), where $\mathbf{I}$ is the identity matrix, $k_0 = 0.2$, and $\nu_0 = 110$. The discrete distributions encoding the priors on objects and orientations, $\theta$ and $\phi$, were uniform over all $V$ and $R$ possibilities. The model and human results clearly exhibit the same qualitative behavior: As the distance between the operator and the left number decreased, the probability the operator took the orientation of the left number increased.

## 6 Experiment 2: Alignment effects on reference frame inference

Our model also predicts that the difference in the number of unambiguous images assigned to the conflicting reference frames should affect the reference frame adopted by the operator image. In this experiment, we test the prediction using the same method as above, but manipulate the number of extra oriented unambiguous objects in each of the competing reference frames (see Figure 5 (a)).

### 6.1 Methods

A total of 80 people participated online through Amazon Mechanical Turk in exchange for $0.20 USD. There were 12 participants who gave an incorrect answer, leaving 68 participants for analysis. The instructions and design were identical to the previous experiment, except that there were two extra factors manipulating the context of the left and right number (5 on the left and 1 on the right or vice versa) and there were only two operator positions (-2 and 2). Figure 5 (a) illustrates example trials of the context manipulations for the operator in position -2.

### 6.2 Results and Discussion

Figure 5 (b) shows that participants were more likely to infer the operator's orientation to be the orientation of whichever side had more objects and it was closer to, replicating the effect of Experiment 1 ($\chi^2(1) = 12.8728, p < 0.0005$). Model results were generated using the same procedure and parameter values as Experiment 1 (except $\nu_0 = 10$ to account for the increased number of objects) and Figure 5 (c) shows its similarity to participant results.

## 7 Conclusions and future directions

In this paper, we introduced the first study of how people infer the reference frame of images in scenes with multiple reference frames. We presented an implicit method for testing reference frame inference, an ideal observer model that predicts people should be sensitive to two scene cues, and

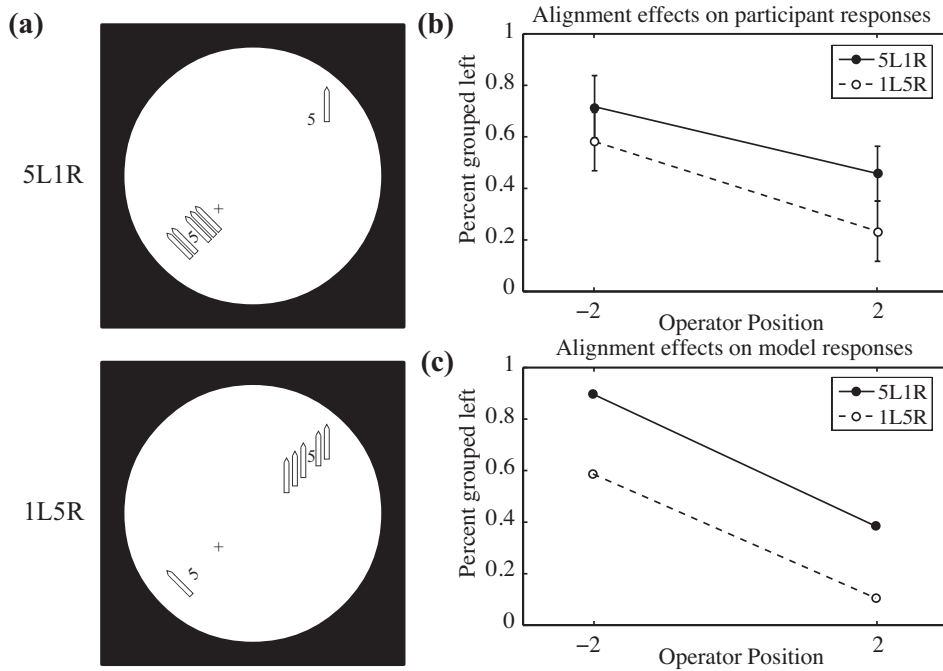

Figure 5: Alignment effects. The operator is more likely to take the orientation of the side with more objects. 5L1R denotes five objects in the left reference frame and one object in the right, and 1L5R indicates the opposite arrangement. (a) Example stimuli, (b) Human results, and (c) Model results.

behavioral evidence supporting its predictions. Because the objects people perceive depend on the orientation of their images in the scene, these results improve our understanding of how the configuration of objects in scenes affects object perception.

We plan to extend our model to capture other cues identified by perceptual psychologists. A first step is to include the bias towards using the up-down axis of the input image [8] by using a non-uniform distribution over rotations (estimating $\theta$). We can capture the *elongation* cue (that the orientation of the spread of images in a scene biases the orientation of the reference frame of the images in the scene [5]) by coupling the covariance matrix ($\Sigma$) and rotation ($r$) of a reference frame. Currently, our model assumes the positions of images in a reference frame are Gaussian distributed; however, people have strong expectations about the arrangement of images in a scene [18]. We plan to compare people's bias to a sophisticated scene segmentation model [19]. We are also interested in cues that depend on the structure of the images or the orientation of the agent in the world, like axes of symmetry [5] or gravitational axes [8].

Another direction for future work is to address an assumption of the model: How do people learn the set of objects and whether or not those objects are orientation-invariant? A potential solution is to combine our model with previous work that presented a nonparametric Bayesian model for learning features and the transformations they are allowed to undergo [20]. Hopefully, incorporating our model into this feature learning method will yield better inferred features and, in turn, will help create better feature generation and object recognition techniques by providing better understanding of how people perceive objects from ambiguous image data.

Finally, we plan to explore how the presented principles scale to more realistic scenes with objects more complex than $+$ and $\times$ and more orientations. Our paradigm provides a principled starting point for investigating how reference frames are identified in scenes with multiple reference frames. It is easily extended to more complex scenes by associating different orientations (or rotations in depth) of an ambiguous image with different arithmetic operators. Our hope is that this leads to a better understanding of object identification and reference frame identification.

**Acknowledgements** We thank Karen Schloss, Stephen Palmer, Anna Rafferty, David Whitney and the Computational Cognitive Science Lab at Berkeley for discussions and AFOSR grant FA-9550-10-1-0232 for support.

## Footnotes

[1]In this paper, we use the following terminology for scene, image, and object. The entire visual input of an observer is a scene. A scene contains a set of images. An image is a part of the visual input that is generated by a single object, which is ambiguous as two or more objects could generate the same image. An object is the item in the world that generates an image in the visual input.

[2]We view the ambiguity of a reference frame as essentially the same as the strength of the intrinsic axes [6].

[3]Though coordinate axes have other properties (e.g., scale), we focus on orientation in this article.

[4]We use slightly different terminology than previous work has done and refer to this principle as alignment rather then symmetry to avoid the ambiguity in the word symmetry (which symmetry we are referring to).

[5]Although we use + and × as the ambiguous images, this method works with any ambiguous images by teaching the participant to use addition in one orientation of the image and multiplication in the other.

# References

[1] D. G. Lowe. Object recognition from local scale-invariant features. In *Proceedings of the International Conference on Computer Vision*, volume 2, pages 1150–1157, 1999.

[2] D. R. Martin, C. C. Fowlkes, and J. Malik. Learning to detect natural image boundaries using local brightness, color, and texture cues. *IEEE Transactions on Pattern Analysis and Machine Intelligence*, 26(5):530–549, 2005.

[3] G. E. Hinton and R. R. Salakhutdinov. Reducing the dimensionality of data with neural networks. *Science*, 313:504–507, 2006.

[4] O. G. Selfridge and U. Neisser. Pattern recognition by machine. In *Computers and thought*, pages 235–267. McGraw-Hill, New York, 1963.

[5] S. E. Palmer. Reference frames in the perception of shape and orientation. In *Object perception: Structure and Process*, pages 121–163. Lawrence Erlbaum Associates, Hillsdale, NJ, 1989.

[6] M. Wiser. The role of intrinsic axes in shape recognition. In *Proceedings of the Third Annual Meeting of the Cognitive Science Society*, pages 184–186, San Mateo, CA, 1981. Morgan Kaufman.

[7] E. Mach. *The analysis of sensations*. Open Court, Chicago, 1914/1959.

[8] I. Rock. *Orientation and form*. Academic Press, New York, 1973.

[9] P. Jolicoeur. The time to name disoriented natural objects. *Memory & Cognition*, 13:289–303, 1985.

[10] M. J. Tarr, P. Williams, W. G. Hayward, and I. Gauthier. Three-dimensional object recognition is viewpoint dependent. *Nature Neuroscience*, 1(4):275–277, 1998.

[11] I. Rock, J. DiVita, and R. Barbeito. The effect on form perception of change of orientation in the third dimension. *Journal of Experimental Psychology: Human Perception and Performance*, 7:719–732, 1981.

[12] S. E. Palmer. What makes triangles point: Local and global effects in configurations of ambiguous triangles. *Cognitive Psychology*, 12:285–305, 1980.

[13] S. E. Palmer. The role of symmetry in shape perception. *Acta Psychologica*, 59:67–90, 1985.

[14] F. Attneave. Triangles as ambiguous figures. *American Journal of Psychology*, 81:447–453, 1968.

[15] S. E. Palmer and N. M. Bucher. Configural effects in perceived pointing of ambiguous triangles. *Journal of Experimental Psychology: Human Perception and Performance*, 7(1):88–114, 1981.

[16] J. Pitman. *Combinatorial Stochastic Processes*. 2002. Notes for Saint Flour Summer School.

[17] R. M. Neal. Markov chain sampling methods for Dirichlet process mixture models. *Journal of Computational and Graphical Statistics*, 9:249–265, 2000.

[18] S. E. Palmer. *Vision Science*. MIT Press, Cambridge, MA, 1999.

[19] E. Sudderth and M. I. Jordan. Shared segmentation of natural scenes using dependent Pitman-Yor processes. In D. Koller, D. Schuurmans, Y. Bengio, and L. Bottou, editors, *Advances in Neural Information Processing Systems 21*, pages 1585–1592. 2009.

[20] J. L. Austerweil and T. L. Griffiths. Learning invariant features using the transformed Indian buffet process. In J. Lafferty, C. K. I. Williams, J. Shawe-Taylor, R.S. Zemel, and A. Culotta, editors, *Advances in Neural Information Processing Systems 23*, pages 82–90. 2010.

